# Identification of Recurrent Patterns in the Activation of Brain Networks

**Firdaus Janoos**[*]   **Weichang Li**   **Niranjan Subrahmanya**
ExxonMobil Corporate Strategic Research
Annandale, NJ 08801

**István Á. Mórocz**   **William M. Wells (III)**
Harvard Medical School
Boston, MA 02115

## Abstract

Identifying patterns from the neuroimaging recordings of brain activity related to the unobservable psychological or mental state of an individual can be treated as a unsupervised pattern recognition problem. The main challenges, however, for such an analysis of fMRI data are: a) defining a physiologically meaningful feature-space for representing the spatial patterns across time; b) dealing with the high-dimensionality of the data; and c) robustness to the various artifacts and confounds in the fMRI time-series.

In this paper, we present a *network-aware feature-space* to represent the states of a general network, that enables comparing and clustering such states in a manner that is a) meaningful in terms of the network connectivity structure; b)computationally efficient; c) low-dimensional; and d) relatively robust to structured and random noise artifacts. This feature-space is obtained from a spherical relaxation of the transportation distance metric which measures the cost of transporting "mass" over the network to transform one function into another. Through theoretical and empirical assessments, we demonstrate the accuracy and efficiency of the approximation, *especially for large problems.*

## 1   Introduction

In addition to functional localization and integration, mapping the neural correlates of "mental states" or "brain states" (*i.e.* the distinct cognitive, affective or perceptive states of the human mind) is an important research topic for understanding the connection between mind and brain [2]. In functional neuroimaging, this problem is equivalent to identifying recurrent spatial patterns from the recorded activation of neural circuits and relating them with the mental state of the subject. Although clustering the data across time to identify the intrinsic state of an individual from EEG and MEG measurements is an established procedure in electrophysiology [19], analysis of temporal patterns in functional MRI data have generally used supervised techniques such as multivariate regression and classification [18, 11, 9], which restrict analysis to observed behavioral correlates of mental state, ignoring any information about the intrinsic mental state that might be present in the data.

In contrast to clustering voxels based on the similarity of their functional activity (*i.e.* along the spatial dimension) [15], the problem of clustering fMRI data along the temporal dimension has not been widely explored in literature, primarily because of the *following challenges*: **a)** Lack of

---

[*]Corresponding Author. firdaus.janoos@exxonmobil.com

a physiologically meaningful metric to compare the difference between the spatial distribution of recorded brain activity (*i.e.* brain states) at two different time-points; **b)** Problems that arise because the number of voxels (*i.e.* dimensions) is orders of magnitude larger ($N \sim \mathcal{O}(10^5)$ vs. $T \sim \mathcal{O}(10^2)$) than the number of scans (*i.e.* samples) ; and **c)** Structured and systematic noise due to factors such as magnetic baseline drift, respiratory and cardiac activity, and head motion. The dimensionality problem in fMRI has been typically addressed through PCA [16], ICA[3] or by selection of a subset of voxels either manually or via regression against the stimulus [18, 11]. PCA has generally been found to be problematic in fMRI [18, 11, 13], since the largest variance principal components usually correspond to motion and physiological noise such as respiration and pulsatile activity, while ICA does not provide an automated way of selecting components. On the other hand, supervised feature-spaces are inherently biased towards the experimental variables against which they were selected or by the investigator's expectations, and may not capture unexpected patterns in the data.

In the *first contribution* of this paper, we address these problems by using a *network-aware metric* that captures the difference between the states $\mathbf{z}_{t_1}$, $\mathbf{z}_{t_2}$ at two different time-points $t_1$, $t_2$ of a temporally evolving function $\mathbf{z}_{\mathbb{G}} : \mathbb{V} \times [0, T] \to \mathbb{R}$ defined on the vertices $\mathbb{V}$ of a network (*i.e.* an weighted undirected graph) $\mathbb{G} = (\mathbb{V}, \mathbb{E})$, in a manner that is *aware of the connectivity structure* $\mathbb{E}$ *of the underlying network*. Intuitively, this network-aware metric assesses the distance between two states $\mathbf{z}_{t_1}$, $\mathbf{z}_{t_2}$ that differ mainly on proximally connected nodes to be less than the distance between states $\mathbf{z}_{t_1}$, $\mathbf{z}_{t_2}$ that differ on unconnected nodes. This concept is illustrated in Fig. 1.

In the context of neuroimaging, where the network measures the *functional connectivity* [4] between brain regions, this implies that two brain activation patterns that differ mainly on functionally similar regions are functionally closer than two that differ on functionally unrelated regions. For example, $\mathbf{z}_{t_1}$ and $\mathbf{z}_{t_2}$ that activated mainly in the cingulo-opercular network would be functionally more similar with each other than with $\mathbf{z}_{t_3}$ that exhibited activity mainly in the fronto-parietal network.

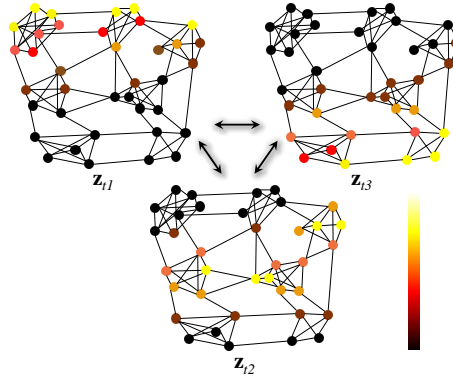

Such network awareness is provided by the Kantorovich metric[20], also called the *transportation distance* (TD), which measures the minimum flow of "mass" over the network to make $\mathbf{z}_{t_1}$ at time $t_1$ match $\mathbf{z}_{t_2}$ that at $t_2$. The cost of this flow is encoded by the weights of the edges of the graph. The Earth Movers Distance (EMD), closely related to the transportation distance, is widely used for clustering and retrieval in computer vision, medical imaging,

Figure 1: Shown are $\mathbf{z}_{t_1}$, $\mathbf{z}_{t_2}$ and $\mathbf{z}_{t_3}$, three states of the function $\mathbf{z}_{\mathbb{G}}$ on the network $\mathbb{G}$. Here, $\mathbf{z}_{t_1}$ and $\mathbf{z}_{t_2}$ activate on more proximal regions of the graph and are hence assessed to be more similar than $\mathbf{z}_{t_1}$ and $\mathbf{z}_{t_3}$. Similarly, for $\mathbf{z}_{t_2}$ and $\mathbf{z}_{t_3}$.

bio-informatics and data-mining [21, 22, 7]. One major strength of this family of metrics for neuroimaging applications, over voxel-wise image matching, is that it allows for partial matches thereby mitigating the effect of small differences between the measurements that arise due to spatial displacement such as head-motion or from random noise [21].

The TD, however, has the *following limitations:* Firstly, it is computationally expensive with worst-case complexity of $\mathcal{O}(N_{\mathbb{V}}^3 \log N_{\mathbb{V}})$ where $N_{\mathbb{V}}$ is the number of nodes in the graph [17]. If the number of time-series observations is $T$, clustering requires $\mathcal{O}(T^2)$ comparisons, making computation prohibitively expensive for large data-sets. Secondly, and more importantly, the metric is the solution to an optimization problem and therefore does not have a tractable geometric structure. For example, there is no closed form expression of the centroid of a cluster under this metric. As a result, determining the statistical properties of clusters obtained under this metric, leave alone developing more sophisticated models, is not straightforward. Although linear embedding (*i.e.* Euclidean) approximations have been proposed for the EMD [12, 22], they are typically defined for comparing probability distributions over regular grids and extension to functions over arbitrary networks is an open problem.

The *second contribution* of this paper is to address these issues through the development of a linear feature-space that provides a good approximation of the transportation distance. This feature–space is motivated by spherical relaxation [14] of the dual polytope of the transportation problem, as described in Section 2. The network function $\mathbf{z}_\mathbb{G}$ is then embedded into an Euclidean space via a *similarity transformation* such that the the transportation distance is well-approximated by the $\ell_2$ distance in this space, as elucidated in Section 3. In contrast to existing linear approximations, the feature-space developed here has *a very simple form* closely related to the *graph Laplacian* [6]. Theoretical bounds to the error the approximation are developed and the accuracy of the method is validated empirically in Section 4.1. Here, we show that the feature–space does not deteriorate, but on the contrary, may improve as the size of the graph increases, making it highly suitable for dealing with large networks like the brain. Its application to extracting the intrinsic mental-states, in an unsupervised manner, from an fMRI study for a visuo-spatial motor task is demonstrated in Section 4.2. Detailed proofs and descriptions are provided in the Supplemental to the manuscript.

## 2 Transportation Distance and Spherical Relaxation

Let $\mathbf{z}_{t_1}$ and $\mathbf{z}_{t_2}$ denote the states of $\mathbf{z}_\mathbb{G}$ at time-points $t_1$, $t_2$ on the graph $\mathbb{G} = (\mathbb{V}, \mathbb{E})$, with nodes $\mathbb{V} = \{1 \dots N_\mathbb{V}\}$ and edges $\mathbb{E} = \{(i,j) \mid i,j \in \mathbb{V}\}$. The symmetric *distance matrix* $W_\mathbb{G}[i,j] \in \mathbb{R}^+$ encodes the cost of transport between nodes $i$ and $j$. Also, define the difference between two states as $\mathbf{dz} = \mathbf{z}_{t_1} - \mathbf{z}_{t_2}$, and assume $\sum_{i \in \mathbb{V}} dz[i] = 0$ without loss of generality [1] . The minimal cost $TD(\mathbf{z}_{t_1}, \mathbf{z}_{t_1})$, of transport $\mathbf{f} : \mathbb{E} \to \mathbb{R}^+$ of "mass" over the network to convert $\mathbf{z}_{t_1}$ into $\mathbf{z}_{t_2}$, is posed as the following linear program (LP):

$$TD(\mathbf{z}_{t_1}, \mathbf{z}_{t_2}) = \min_{\mathbf{f}} \sum_{i \in \mathbb{V}} \sum_{b \in \mathbb{V}} f[i,j] W_\mathbb{G}[i,j], \qquad \text{subject to} \quad \sum_j f[i,j] - \sum_j f[j,i] = dz[i]. \quad (1)$$

The corresponding TP dual, formulated in the unrestricted dual variables $\mathbf{g} : \mathbb{V} \to \mathbb{R}$, is:

$$TD(\mathbf{z}_{t_1}, \mathbf{z}_{t_2}) = \max_{\mathbf{g}} \langle \mathbf{g}, \mathbf{dz} \rangle \qquad \text{subject to} \qquad A\mathbf{g} \le \mathbf{w}_\mathbb{G} \qquad (2)$$

$$\text{where} \quad A = \begin{bmatrix} 1 & -1 & 0 & \dots & 0 \\ 1 & 0 & -1 & \dots & 0 \\ \vdots & & & \ddots & \vdots \\ 1 & 0 & 0 & \dots & -1 \\ -1 & 1 & 0 & \dots & 0 \\ 0 & 1 & -1 & \dots & 0 \\ \vdots & & & \ddots & \vdots \\ 0 & 1 & 0 & \dots & -1 \\ \vdots & & & \ddots & \vdots \end{bmatrix} \quad \text{and} \quad \mathbf{w}_\mathbb{G} = \begin{pmatrix} W_\mathbb{G}[1,2] \\ W_\mathbb{G}[1,3] \\ \vdots \\ W_\mathbb{G}[1,N] \\ W_\mathbb{G}[2,1] \\ W_\mathbb{G}[2,3] \\ \vdots \\ W_\mathbb{G}[2,N] \\ \vdots \end{pmatrix} .$$

The feasible set of the dual is a convex polytope formed by the intersection of the half-spaces specified by the constraints $\{\mathbf{a}_{i,j}, i = 1 : N_\mathbb{V}, j = 1 \dots N_\mathbb{V}, i \ne j\}$, corresponding to the rows of A, and containing a $+1$ entry in the $i$–th position and a $-1$ entry in the $j$–th position. These constraints which form normals to the hyper-planes bounding this polytope, are symmetrically distributed in the $+i \times -j$ quadrant of $\mathbb{R}^{N_\mathbb{V}}$ for each combination of $i$ and $j$ . Moreover, A is totally uni-modular [5], and has rank of $N_\mathbb{V} - 1$ with the LP polytope lying in an $N_\mathbb{V} - 1$ dimensional space orthogonal to $\mathbb{1}_{N_\mathbb{V}}$, the $\mathbb{1}$ –vector in $\mathbb{R}^{N_\mathbb{V}}$. In the discussion below, we operate in the original $\mathbb{R}^{N_\mathbb{V}}$ notation, by considering its restriction to the $N_\mathbb{V} - 1$ dimensional sub-space $\{\mathbf{g} \in \mathbb{R}^{N_\mathbb{V}} \mid \langle \mathbf{g}, \mathbb{1}_{N_\mathbb{V}} \rangle = 0\}$, *i.e.* $\sum_{i \in \mathbb{V}} g[i] = 0$. The optimal solution to this problem will lie on the $N_\mathbb{V} - 1$ simplicial complex formed by intersections of the $N_\mathbb{V} - 1$ dimensional hyper-planes each at a distance of $W_\mathbb{G}[i,j]/\sqrt{2}$ from the origin, and in the non-degenerate case will coincide with the extreme-points of the polytope $A\mathbf{g} \le \mathbf{w}_\mathbb{G}$.

Consider the a special case for the fully-connected graph with $W_\mathbb{G}[i,j] = 1, \forall i,j \in \mathbb{V}$. Here,

$$TD(\mathbf{z}_{t_1}, \mathbf{z}_{t_2}) = \max_{\mathbf{g}} < \mathbf{g}, \mathbf{dz} > \qquad \text{subject to} \qquad A\mathbf{g} \le \mathbb{1}_{N_\mathbb{V} \times (N_\mathbb{V}-1)}. \qquad (3)$$

Each hyper-plane of the LP polytope is at distance $1/\sqrt{2}$ from the origin and the maximum inscribed hyper-sphere, with center at the origin and radius $1/\sqrt{2}$ touches all the polytope's hyper-planes. *The*

*main idea* of the embedding is to use the regularity of this polytope, with $2^{N_{\mathbb{V}}} - 2$ extreme points symmetrically distributed in $\mathbb{R}^{N_{\mathbb{V}}-1}$ (§ Proposition 2 in the Supplemental) and approximate it by this hyper-sphere. Relaxing the feasible set of the TP dual from the convex polytope to this hyper-sphere, eqn. (2) becomes:

$$\widehat{\mathrm{TD}}(\mathbf{z}_{t_1}, \mathbf{z}_{t_2}) = \max_{\mathbf{g}} <\mathbf{g}, \mathbf{dz}> \qquad \text{such that} \qquad ||\mathbf{g}||_2 = \frac{1}{\sqrt{2}}, \tag{4}$$

which has a direct solution

$$\widehat{\mathrm{TD}}(\mathbf{z}_{t_1}, \mathbf{z}_{t_2}) = \frac{1}{\sqrt{2}}||\mathbf{dz}|| = \frac{1}{\sqrt{2}}||\mathbf{z}_{t_1} - \mathbf{z}_{t_2}|| \qquad \text{with} \qquad \widehat{\mathbf{g}}^* = \frac{1}{\sqrt{2}}\frac{\mathbf{dz}}{||\mathbf{dz}||} \tag{5}$$

The worst-case error of this approximation is $\mathcal{O}(||\mathbf{dz}||)$ (§Theorem 1 of the Supplemental), proving that quality of the linear approximation for a graph where all nodes are equidistant neighbors of each other *does not deteriorate* as the size of the graph increases.

## 3 Linear Feature Space Embedding

In the case of an arbitrary distance matrix $W_{\mathbb{G}}$, however, the polytope loses its regular structure, and has a variable number of extreme points. Also, in general, the maximal inscribed hyper-sphere does not touch all the bounding hyper-planes, resulting in a very poor approximation [14]. Therefore, to use the spherical relaxation for the general problem, we apply a *similarity transformation* M, such that $A \cdot M = \mathrm{diag}\{\mathbf{w}_{\mathbb{G}}\}^{-1}A$ and M positive semi-definite. Expressing eqn. (2) in terms of a new variable $\xi \triangleq M\mathbf{g}$, we see that the general problem:

$$\mathrm{TD}(\mathbf{z}_{t_1}, \mathbf{z}_{t_2}) = \max_{\mathbf{g}} <\mathbf{g}, \mathbf{dz}> \qquad \text{such that} \qquad A\mathbf{g} \leq \mathbf{w}_{\mathbb{G}} \tag{6}$$

is equivalent to the special case given by eqn. (3), in a transformed space, as per:

$$\mathrm{TD}(\mathbf{z}_{t_1}, \mathbf{z}_{t_2}) = \max_{\xi} <M^-\xi, \mathbf{dz}> \qquad \text{such that} \qquad A\xi \leq \mathbb{1}_{N_{\mathbb{V}} \times (N_{\mathbb{V}}-1)}, \tag{7}$$

where $M^-$ is the (pseudo-)inverse of M. Then, the approximation of eqn. (4) yields: $\widehat{\mathrm{TD}}(\mathbf{z}_{t_1}, \mathbf{z}_{t_2}) = \frac{1}{\sqrt{2}}||M^{-1^\top}(\mathbf{z}_{t_1} - \mathbf{z}_{t_2})||$.

As shown in Supplemental Section A, the transformation matrix $M = \frac{1}{N_{\mathbb{V}}}\mathcal{L}_{\mathbb{G}}$, where $\mathcal{L}_{\mathbb{G}} = D_{\Delta_{\mathbb{G}}} - \Delta_{\mathbb{G}}$ is the *un-normalized Laplacian matrix* of the graph. Here, $\Delta_{\mathbb{G}}$ is the *adjacency matrix* such that $\Delta_{\mathbb{G}}[i,j] = W_{\mathbb{G}}[i,j]^{-1}, \forall i \neq j$ and $D_{\Delta_{\mathbb{G}}}$ is the diagonal degree matrix with $D_{\Delta_{\mathbb{G}}}[i,i] = \sum_{j\in\mathbb{V}}\Delta_{\mathbb{G}}[i,j]$ and $D_{\Delta_{\mathbb{G}}}[i,j] = 0$, for $i \neq j$. Defining $V\Lambda V^\top = \mathcal{L}_{\mathbb{G}}$ as the eigen-system of the graph Laplacian, and the projection of $\mathbf{z}_t$ onto the feature space $V\Lambda^-$ as $\widehat{\mathbf{z}}_t = \Lambda^- V^\top \mathbf{z}_t$ yields:

$$\widehat{\mathrm{TD}}(\mathbf{z}_{t_1}, \mathbf{z}_{t_2}) = \frac{1}{\sqrt{2}}||\Lambda^- V^\top \mathbf{dz}|| = \frac{1}{\sqrt{2}}||\widehat{\mathbf{z}_{t_1}} - \widehat{\mathbf{z}_{t_2}}||. \tag{8}$$

Consequently, the transportation distance can be approximated by a $\ell_2$ metric through a similarity transformation of the original space. In this case the error of the approximation is $\mathcal{O}(\lambda_{\min}^{-1}||\mathbf{dz}||_2)$ (§Theorem 1 of the Supplemental), which implies that the *approximation improves* as the smallest eigenvalue of the graph Laplacian increases. Also, notice that the eigenvector $\mathbf{v}_{N_{\mathbb{V}}}$ of $\mathcal{L}_{\mathbb{G}}$ corresponding to the smallest eigenvalue $\lambda_{N_{\mathbb{V}}} = 0$ is a constant vector, and therefore $\langle \mathbf{v}_{N_{\mathbb{V}}}, \mathbf{dz}\rangle = 0$ by the requirement that $\sum_{i\in\mathbb{V}} dz[i] = 0$, thereby automatically reducing the dimension of the projected space to $N_{\mathbb{V}} - 1$.

Dimensionality reduction of the feature-space can be achieved by discarding eigenvectors of $\mathcal{L}_{\mathbb{G}}$ with the $P$ largest eigenvalues whose inverse sum contributes to less than a certain percentage of the total inverse spectral energy. If eigenvectors with eigenvalues $\lambda_1 \geq \lambda_2 \geq \ldots \geq \lambda_P$ are discarded, the additional error in $\widehat{\mathrm{TD}}(\mathbf{z}_{t_1}, \mathbf{z}_{t_2})$ is equal to $\sqrt{\sum_{k=1}^{P}\lambda_k^{-2}}/\sqrt{\sum_{k=P+1}^{N_{\mathbb{V}}}\lambda_k^{-2}}$.

## 4 Results

First, we start by providing an empirical validation of the approximation to the transportation distance in Section 4.1 And then the feature-space is used to find representative patterns (*i.e.* brain states) in the dynamically changing activations of the brain during a visuo-motor task in Section 4.2.

### 4.1  Validation

To validate the linear approximation to the transportation distance on networks, like the brain, that exhibit a scale-free property [1], we simulated random graphs of $N_\mathbb{V}$ vertices using the following procedure: **a)** Create an edge between nodes $i$ and $j$ with probability $\propto \beta(d_i + d_j + \epsilon)$, where $d_i$ is the degree of node $i$, and $\beta$, $\epsilon$ are constants that are varied across experiments; **b)** sample the weight of the edge from a $\chi_1^2$ distribution scaled by a constant $\gamma$, varied across experiments. For each instance $\mathbb{G}^{(n)}$ of the graph, a set of $T = 100$ states $\mathbf{z}_t : \mathbb{V}^{(i)} \to \mathbb{R}$, $t = 1 \ldots 10^4$ were sampled from a standard normal distribution such that $\sum_i dz[i] = 0$. The experiment was repeated 10 times at graph sizes of $N_\mathbb{V} = 2^n$, $n = 4 \ldots 12$.

The transportation problem was solved using network simplex [17] in the IBM CPLEX® optimization package, while the linear approximation was implemented in Matlab®. All experiments were run on a 2.6Hz Opteron cluster with 16 processors and 32GB RAM each. The amortized running time for one pair-wise comparison is shown in Fig. 2(a). While an individual run of the network simple algorithm is much faster than the eigen-system computation of the linear feature-space, repeatedly solving $\mathrm{TD}(\mathbf{z}_{t_1}, \mathbf{z}_{t_2})$ for all pairs of $\mathbf{z}_{t_1}, \mathbf{z}_{t_2}$ is orders of magnitude slower than a simple Euclidean distance, reducing its net efficiency.

The relative error, as shown in Fig. 2(b), reduces with increasing number of vertices, approximately as $\mathcal{O}(N_\mathbb{V}^{-1})$. This is because the approximation error for an arbitrary graph is $\mathcal{O}(\lambda_{\min}^{-1}\|\mathbf{dz}\|_2)$, while for random graphs satisfying basic regularity conditions the eigenvalues of the graph Laplacian increase as $\mathcal{O}(N_\mathbb{V})$ [8]. In comparison, the Euclidean metric $\|\mathbf{z}_{t_1} - \mathbf{z}_{t_2}\|_2$ starts with a much higher relative error with respect to the transportation distance, and although its error also reduces with graph size, the trend is slower. Secondly, the variance of the error is much higher than the linear embedding proposed here.

In the context of clustering, which is the motivation for this work, a more important property is that the approximation preserve the relative configuration (*i.e.* homomorphism) between observations rather than the numerical values of their distances (*i.e.* isomorphism), as characterized by its ability to preserve the relative ordering between points (*i.e.* a *topological equivalence* property). From Fig. 2(c), we observe that for data-points that are relatively close to each other, the ordering relationships are preserved with very high accuracy and it reduces as the relative distance between the points increases.

Another important property for an embedding scheme, especially for non-linear manifolds like that induced by the TD, is its ability to preserve the relative distances between points that are in local neighborhoods (*i.e.* a *coordinate chart* property ). This is quantified by a *normalized neighborhood error* as defined by:

$$\mathrm{NormErr}(\mathbf{z}_{t_1}, \mathbf{z}_{t_2}) = \frac{|a - b|}{|a|}, \text{where } a = \frac{\mathrm{TD}(\mathbf{z}_{t_1}, \mathbf{z}_{t_2})}{\sum_{n \in \mathcal{N}_{t_1}} \mathrm{TD}(\mathbf{z}_{t_1}, \mathbf{z}_{t_n})} \text{ and } b = \frac{\widehat{\mathrm{TD}}(\mathbf{z}_{t_1}, \mathbf{z}_{t_2})}{\sum_{n \in \mathcal{N}_{t_1}} \widehat{\mathrm{TD}}(\mathbf{z}_{t_1}, \mathbf{z}_{t_n})}.$$

The neighborhoods $\mathcal{N}_{t_1}$ contain the 10 nearest neighbors of $\mathbf{z}_{t_1}$ under the $\mathrm{TD}$ and $\widehat{\mathrm{TD}}$ metrics respectively. The formulation has the effect of normalizing the distance between $\mathbf{z}_{t_1}, \mathbf{z}_{t_2}$ with respect to the local neighborhood of $\mathbf{z}_{t_1}$. It can be seen in Fig. 2(d) that the approximation error according to this measure is extremely low and almost constant with respect to $N_\mathbb{V}$ for points that are close to each other. These plot indicate that although $\widehat{\mathrm{TD}}$ does not hold for distant points on the manifold induced by TD, it provides a good approximation of its topology.

### 4.2  Neuroimaging Data

Clustering using the feature-space described in this paper was applied to a data-set of fifteen subjects performing a visuo-motor task during functional MR imaging to discover salient patterns of recurrent brain activation. The subjects were visually exposed to oriented wedges filled with high-contrast random noise patterns and displayed *randomly* in one of four quadrants. They were asked to focus on a center dot and to perform a finger-tapping motion with the right or left hand when the visual wedge was active in the upper right or lower left quadrants, respectively. Block length of each visual wedge stimulation varied from 5 to 15s and noise patterns changed at a frequency of 5Hz. A multi-shot 3D Gradient Echo Planar Imaging (EPI) sequence accelerated in the slice encoding direction with GRAPPA and UNFOLD was used on a GE 3T MRI scanner with a quadrature head

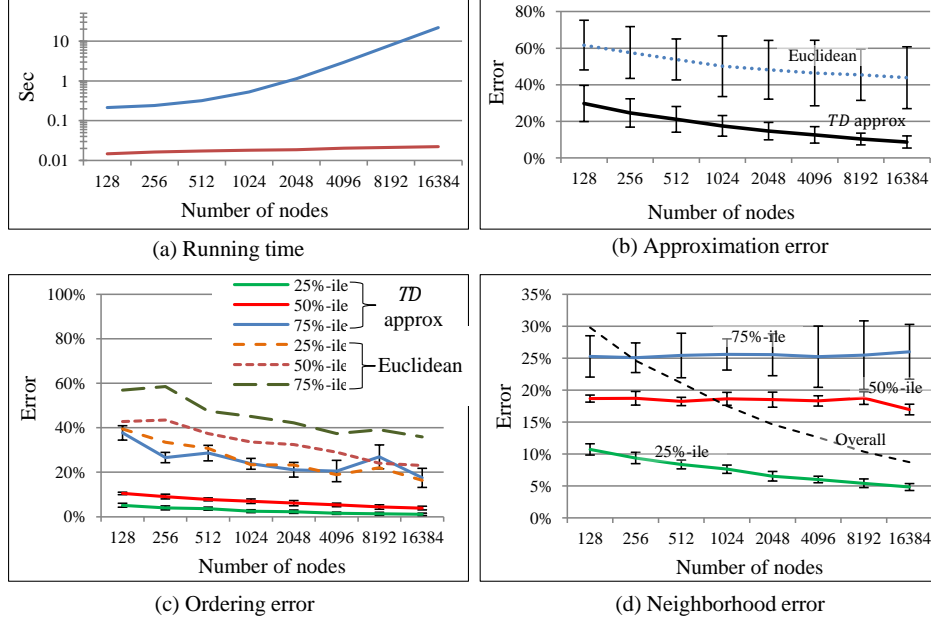

(a) Running time

(b) Approximation error

(c) Ordering error

(d) Neighborhood error

Figure 2: **Fig.(a)** shows the (amortized) per-comparison running time in seconds for the transportation distance TD and its approximation $\widehat{\text{TD}}$ with respect to with respect to graph size $N_{\mathbb{V}}$. In **Fig.(b)** the relative approximation error $(\widehat{\text{TD}} - \text{TD})/\text{TD}$ ($\pm 1$ std.dev.) is graphed. The error for an Euclidean approximation $\|\mathbf{z}_{t_1} - \mathbf{z}_{t_2}\|_2$ is also shown for comparison. **Fig.(c)** shows the quartile-wise ordering error ($\pm 1$ std.dev.). For each $\mathbf{z}_{t_1}$, the fraction of $\{\mathbf{z}_{t_2}, t_2 = 1 \ldots T, t_2 \neq t_1\}$ that are misordered by $\widehat{\text{TD}}(\mathbf{z}_{t_1}, \mathbf{z}_{t_2})$ with respect to the ordering induced by $\text{TD}(\mathbf{z}_{t_1}, \mathbf{z}_{t_2})$ is calculated. The set $\{\mathbf{z}_{t_2}\}$ is divided into quartiles according to their distance $\text{TD}(\mathbf{z}_{t_1}, \mathbf{z}_{t_2})$ from $\mathbf{z}_{t_1}$, where the 25 percentile is set of the first 25% closest points to $\mathbf{z}_{t_1}$ (similarly for the 50 and 75%-iles). Also shown is the ordering error of the Euclidean metric with respect to TD. Error-bars are omitted for clarity. **Fig (d)** shows the quartile-wise approximation error normalized by the average distance of its 10 nearest neighbors. The dashed line shows the un-normalized approximation error (§ Fig.(b)) for reference.

coil and $T = 171$ volumes were acquired at TR=1.05s, an isotropic resolution of 3mm, with total imaging time of 3min and the first five volumes were discarded from the analysis. High resolution anatomical scans were also acquired, bias-field corrected, normalized to an MNI atlas space and segmented into gray and white matter regions. The fMRI scans were motion corrected using linear registration and co- registered with the structural scans using SPM8 [16]. Next, the time-series data were high-pass filtered (0.5Hz) to remove gross artifacts due to breathing, blood pressure changes and scanner drift. The data were first analyzed for task related activity using a general linear model (GLM) with SPM8 for reference. The design matrix included a regressor for the presentation of the wedge in each quadrant, convolved with a canonical hemodynamic response function. These results are shown in Fig. 3(a).

Note that the data for each subject were processed separately. The mean volume of the time-series was then subtracted, white matter masked out and all further processing was performed on the gray matter. The functional networks for a subject were computed by estimating the correlations between voxels using the method described in Supplemental Section C, that is sparse, consistent and computationally efficient. The distance matrix of the functional connectivity graph was constructed as $W_{\mathbb{G}}[i, j] = -\log(|\rho[i, j]|/\tau)$, where $\rho[i, j]$ is the correlation between voxels $i$ and $j$ and $\tau$ is a user-defined scale parameter (typically set to 10). This mapping has the effect that $W_{\mathbb{G}}[i, j] \to 0$ as $|\rho[i, j]| \to 1$ and $W_{\mathbb{G}}[i, j] \to \infty$ as $|\rho[i, j]| \to 0$.

The linear feature-space (§eqn. (8)) was computed from the graph Laplacian of $\Delta_{\mathbb{G}}$, where $\Delta_{\mathbb{G}}[i, j] = W_{\mathbb{G}}[i, j]^{-1}$, retaining only those basis vectors corresponding to the top 80 eigenvalues ($\approx 50\%$ of the spectral energy), and the fMRI volumes were embedded into this low dimensional space. For

clustering, the state-space method (SSM) of Janoos, *et al.* [13] was used, which is a modified hidden Markov model with Gaussian emission probabilities that assigns a state (*i.e.* cluster) label to each scan while accounting for the temporal blurring cause by the hemodynamic response. This method associates each time-point $t$ of the fMRI time-series with a vector $\pi_t = \{\pi_t[1] \dots \pi_t[K] \mid \pi_t[k] \in [0,1], \sum_k \pi_t[k] = 1\}$ giving the probability of belonging to state $1 \dots K$. A multinomial logistic classifier (MLC) was then trained to predict the wedge position at time $t$ from $\pi_t$. The number of clusters was determined by selecting a value of $5 \leq K \leq 15$ that minimized the generalization error of the MLC, which acts as a statistic to assess the quality of the model-fit and perform model selection.

It should be noted here that identification of patterns of recorded brain activity was performed in a purely unsupervised manner. Only model selection and model interpretation was done, *post hoc*, using observable correlates of the unobservable mental state of the subject. Spatial maps for each wedge orientation were computed as an average of cluster centroids weighted by the MLC weights for that orientation. The $z$-statistic spatial maps for the group from this analysis are shown in Fig. 3(b), and exhibit the classic contra-lateral retinotopic organization of the primary visual cortex with the motor representation areas in both hemispheres. Fig. 3(c) shows the distribution of state probabilities for one subject corresponding to a sequence of wedges oriented in each quadrant for $4 \times$TRs each. Here, we see that the probability of a particular state is highly structured with respect to the orientation of the wedge. For example, at the start of the presentation with the wedge in the lower-right quadrant, state 1 is most probable. But by the second interval, state 2 becomes more dominant and this distribution remains stable for the rest of this presentation. Then, as the display transitions to the lower-left quadrant, states 3 and 4 become equiprobable. However, as this orientation is maintained, the probability distribution peaks about state 4 and remains stable. A similar pattern in observed in the probability distributions for the other orientations.

For comparison, we also performed the same clustering using a low-dimensional PCA basis explaining $\approx 50\%$ of the variance of the data ($d = 60$), and the low-dimensional basis (CorrEig) proposed by [13] derived from the eigen-decomposition of the voxel-wise correlation matrix ($d \approx 110$). Multinomial logistic classifiers (MLC) were trained for each case and number of states were tuned using the same procedure as above. The spatial maps reconstructed from these two feature-spaces (not shown here) exhibited task-specific activation patterns, although the foci were much weaker and much more diffused as compared to those of the $\widehat{TD}$ feature-space. The error of the MLC in predicting the stimulus at time $t$ from the state probability vector $\pi_t$, which reflects the model's ability to capture patterns in the data related to the mental state of the subject, for these three feature spaces is listed in Table 1.

|  | Lower right | Lower left | Upper left | Upper right | Overall |
|---|---|---|---|---|---|
| $\widehat{TD}$ | 0.17 ($\pm$ 0.05) | 0.13 ($\pm$ 0.02) | 0.21 ($\pm$ 0.04) | 0.12 ($\pm$ 0.03) | 0.16 ($\pm$ 0.07) |
| PCA | 0.41 ($\pm$ 0.08) | 0.37 ($\pm$ 0.10) | 0.39 ($\pm$ 0.09) | 0.36 ($\pm$ 0.08) | 0.38 ($\pm$ 0.18) |
| CorrEig | 0.29 ($\pm$ 0.05) | 0.22 ($\pm$ 0.04) | 0.30 ($\pm$ 0.06) | 0.23 ($\pm$ 0.05) | 0.26 ($\pm$ 0.10) |

Table 1: The generalization error of the multinomial logistic classifier to predict the orientation of the wedge from the distribution of state labels estimated by the SSM trained on three low-dimensional representations of the fMRI data: **a)** the approximate transportation distance $\widehat{TD}$; **b)** PCA basis; and **c)** the eigen basis of the voxel-wise correlation matrix (CorrEig). Due to the random presentation of wedge orientations, the chance level prediction error varied between $68\% - -81\%$ for each subject.

We see that the prediction error – and therefore the ability of the state-space model to identify mental-state related patterns in the data – is significantly better for the $\widehat{TD}$ feature-space as compared to that of [13] ($p < 10^{-6}$, 1-sided 2-sample $t$-test) while PCA performs significantly worse than both the other feature-spaces, as expected. Moreover, the $\widehat{TD}$ representation provides a significant difference ($p < 0.001$, 1-sided 2-sample $t$-test) between the prediction rates for the wedge orientations with and without the finger-tapping task, implying that the model is able to better detect brain patterns when both visual and motor regions are involved as compared to those involving only the visual regions, probably because of the more distinct functional signature of the former.

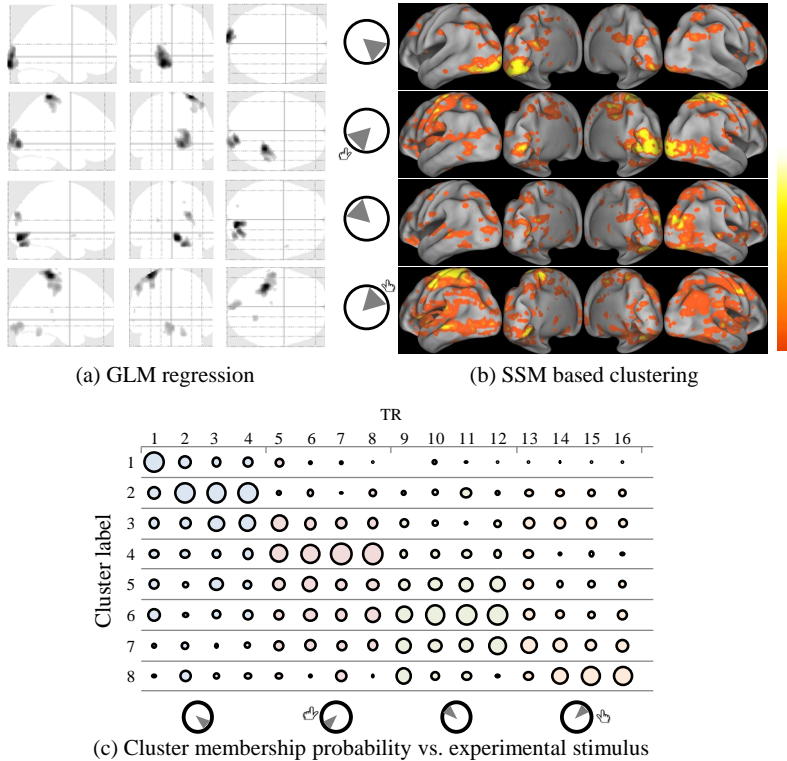

(a) GLM regression

(b) SSM based clustering

(c) Cluster membership probability vs. experimental stimulus

Figure 3: **Fig. (a)**: Group-level maximum intensity projections of significantly activated voxels ($p < 0.05$, FWE corrected) at the four orientations of the wedge and the hand motor actions, computed using SPM8 **Fig. (b)**: Group-level $z$-maps showing the activity for each orientation of the wedge computed as an average of cluster centroids weighted by the MLC weights. Displayed are the posterio-lateral and posterio-medial views of the left and right hemispheres respectively. Values $|z| \leq 1$ have been masked out for visual clarity. **Fig. (c)**: The SSM state probability vector $\pi_t$ for one subject. The size of the circles corresponds to the marginal probability $\pi_t[k]$ of state $k = 1 \ldots 8$ during the display of the wedge in lower right, lower left, upper left and upper right quadrants for 4TRs each. States have been relabeled for expository purposes.

## 5   Conclusion

In this paper, we have presented an approach to compare and identify patterns of brain activation during a mental process using a distance metric that is aware of the connectivity structure of the underlying brain networks. This distance metric is obtained by an Euclidean approximation of the transportation distance between patterns via a spherical relaxation of the linear-programming dual polytope. The embedding is achieved by a transformation of the original space of the function with the graph Laplacian of the network. Intuitively, the eigen-system of graph Laplacian indicates min-flow / max-cut partitions of the graph [10], and therefore projecting on these basis increases the cost if the difference between two states of the function is concentrated on relatively distant or disconnected regions of the graph.

We provided theoretical bounds on the quality of the approximation and through empirical validation demonstrated low error that, importantly, decreases as the size of the problem increases. We also showed the superior ability of this distance metric to identify salient patterns of brain activity, related to the internal mental state of the subject, from an fMRI study of visuo-motor tasks.

The framework presented here is applicable to the more general problem of identifying patterns in time-varying measurements distributed over a network that has an intrinsic notion of distance and proximity, such as social, sensor, communication, transportation, energy and other similar networks. Future work would include assessing the quality of the approximation for sparse, restricted topology, small-world and scale-free networks that arise in many real world cases, and applying the method for detecting patterns and outliers in these types of networks.

## Footnotes

[1] Add dummy node with index $N_\mathbb{V} + 1$ where $dz[N_\mathbb{V}+1] = -\sum_{i \in \mathbb{V}} dz[i]$ and $W_\mathbb{G}[i, N_\mathbb{V}+1] = 0, \forall i \in \mathbb{V}$.

# References

[1] Achard, S., Salvador, R., Whitcher, B., Suckling, J., Bullmore, E.: A resilient, low-frequency, small-world human brain functional network with highly connected association cortical hubs. Neurosci 26(1), 63–72 (Jan 2006) 5

[2] Barrett, L.F.: The future of psychology: Connecting mind to brain. Perspect Psychol Sci 4(4), 326–339 (Jul 2009) 1

[3] Calhoun, V.D., Adali, T., Pearlson, G.D., Pekar, J.J.: Spatial and temporal independent component analysis of functional MRI data containing a pair of task-related waveforms. Hum Brain Map 13(1), 43–53 (May 2001) 2

[4] Cecchi, G., Rish, I., Thyreau, B., Thirion, B., Plaze, M., Paillere-Martinot, M.L., Martelli, C., Martinot, J.L., Poline, J.B.: Discriminative network models of schizophrenia. In: Adv Neural Info Proc Sys (NIPS) 22, pp. 252–260 (2009) 2

[5] Chandrasekaran, R.: Total unimodularity of matrices. SIAM Journal on Applied Mathematics 17(6), pp. 1032–1034 (1969) 3

[6] Chung, F.: Lectures on Spectral Graph Theory. CBMS Reg Conf Series Math, Am Math Soc (1997) 3

[7] Deng, Y., Du, W.: The Kantorovich metric in computer science: A brief survey. Electronic Notes in Theoretical Computer Science 253(3), 73 – 82 (2009) 2

[8] Ding, X., Jiang, T.: Spectral distributions of adjacency and Laplacian matrices of random graphs. The Annals of Applied Probability 20(6), 2086 –2117 (2010) 5

[9] Friston, K., Chu, C., Mouro-Miranda, J., Hulme, O., Rees, G., Penny, W., Ashburner, J.: Bayesian decoding of brain images. Neuroimage 39(1), 181–205 (Jan 2008) 1

[10] Grieser, D.: The first eigenvalue of the laplacian, isoperimetric constants, and the max flow min cut theorem. Archiv der Mathematik 87, 75–85 (2006) 8

[11] Haynes, J.D., Rees, G.: Decoding mental states from brain activity in humans. Nature Rev: Neurosci 7(7), 523–534 (Jul 2006) 1, 2

[12] Indyk, P., Thaper, N.: Fast color image retrieval via embeddings. ICCV (2003) 2

[13] Janoos, F., Singh, S., Wells III, W., Mórocz, I.A., Machiraju, R.: State–space models of mental processes from fMRI (2011) 2, 7

[14] Khachiyan, L.G., Todd, M.J.: On the complexity of approximating the maximal inscribed ellipsoid for a polytope. Mathematical Programming 61, 137–159 (1993) 3, 4

[15] Lashkari, D., Sridharan, R., Golland, P.: Categories and functional units: An infinite hierarchical model for brain activations. In: Advances in Neural Information Processing Systems. vol. 23, pp. 1252–1260 (2010) 1

[16] Multiple: Statistical Parametric Mapping: The Analysis of Functional Brain Images. Acad Press (2007) 2, 6

[17] Orlin, J.B.: On the simplex algorithm for networks and generalized networks. In: Mathematical Programming Essays in Honor of George B. Dantzig Part I, Mathematical Programming Studies, vol. 24, pp. 166–178. Springer Berlin Heidelberg (1985) 2, 5

[18] O'Toole, A.J., Jiang, F., Abdi, H., Pnard, N., Dunlop, J.P., Parent, M.A.: Theoretical, statistical, and practical perspectives on pattern-based classification approaches to the analysis of functional neuroimaging data. J Cog Neurosci 19(11), 1735–1752 (Nov 2007) 1, 2

[19] Pascual-Marqui, R.D., Michel, C.M., Lehmann, D.: Segmentation of brain electrical activity into microstates: model estimation and validation. IEEE Trans Biomed Eng 42(7), 658–665 (Jul 1995) 1

[20] Rachev, S.T., Ruschendorf, L.: Mass transportation problems: Volume I: Theory (probability and its applications) (March 1998) 2

[21] Rubner, Y., Tomasi, C., Guibas, L.J.: The earth mover"s distance as a metric for image retrieval (1998) 2

[22] Shirdhonkar, S., Jacobs, D.: Approximate earth mover's distance in linear time. In: Comp Vis Pat Recog., IEEE Conf. pp. 1 –8 (23-28 2008) 2

